# An Optimization Method of Layered Neural Networks based on the Modified Information Criterion

**Sumio Watanabe**

Information and Communication R & D Center
Ricoh Co., Ltd.
3-2-3, Shin-Yokohama, Kohoku-ku, Yokohama, 222 Japan
sumio@ipe.rdc.ricoh.co.jp

## Abstract

This paper proposes a practical optimization method for layered neural networks, by which the optimal model and parameter can be found simultaneously. We modify the conventional information criterion into a differentiable function of parameters, and then, minimize it, while controlling it back to the ordinary form. Effectiveness of this method is discussed theoretically and experimentally.

## 1 INTRODUCTION

Learning in artificial neural networks has been studied based on a statistical framework, because the statistical theory clarifies the quantitative relation between the empirical error and the prediction error. Let us consider a function $\varphi(w; x)$ from the input space $\mathbf{R}^K$ to the output space $\mathbf{R}^L$ with a parameter $w$. We assume that training samples $\{(x_i, y_i)\}_{i=1}^N$ are taken from the true probability density $Q(x, y)$. Let us define the empirical error by

$$E_{emp}(w) = \frac{1}{N} \sum_{i=1}^N \|y_i - \varphi(w; x_i)\|^2, \tag{1}$$

and the prediction error by

$$E(w) = \int\int \|y - \varphi(w; x)\|^2 Q(x, y) dx dy. \qquad (2)$$

If we find a parameter $w^*$ which minimizes $E_{emp}(w)$, then

$$< E(w^*) > = (1 + \frac{2(F(w^*) + 1)}{NL}) < E_{emp}(w^*) > + o(\frac{1}{N}), \qquad (3)$$

where $< \cdot >$ is the average value for the training samples, $o(1/N)$ is a small term which satisfies $No(1/N) \to 0$ when $N \to \infty$, and $F(w^*)$, $N$, and $L$ are respectively the numbers of the effective parameters of $w^*$, the training samples, and output units.

Although the average $< \cdot >$ cannot be calculated in the actual application, the optimal model for the minimum prediction error can be found by choosing the model that minimizes the Akaike information criterion (AIC) [1],

$$J(w^*) = (1 + \frac{2(F(w^*) + 1)}{NL}) E_{emp}(w^*). \qquad (4)$$

This method was generalized for arbitrary distance [2]. The Bayes information criterion (BIC) [3] and the minimum description length (MDL) [4] were proposed to overcome the inconsistency problem of AIC that the true model is not always chosen even when $N \to \infty$.

The above information criteria have been applied to the neural network model selection problem, where the maximum likelihood estimator $w^*$ was calculated for each model, and then information criteria were compared. Nevertheless, the practical problem is caused by the fact that we can not always find the maximum likelihood estimator for each model, and even if we can, it takes long calculation time.

In order to improve such model selection procedures, this paper proposes a practical learning algorithm by which the optimal model and parameter can be found simultaneously. Let us consider a modified information criterion,

$$J_\alpha(w) = (1 + \frac{2(F_\alpha(w) + 1)}{NL}) E_{emp}(w). \qquad (5)$$

where $\alpha > 0$ is a parameter and $F_\alpha(w)$ is a $C^1$-class function which converges to $F(w)$ when $\alpha \to 0$. We minimize $J_\alpha(w)$, while controlling $\alpha$ as $\alpha \to 0$, To show effectiveness of this method, we show experimental results, and discuss the theoretical background.

## 2    A Modified Information Criterion

### 2.1    A Formal Information Criterion

Let us consider a conditional probability distribution,

$$P(w, \sigma; y|x) = \frac{1}{(2\pi\sigma^2)^{L/2}} \exp(-\frac{\|y - \varphi(w; x)\|^2}{2\sigma^2}), \qquad (6)$$

where a function $\varphi(w; x) = \{\varphi_i(w; x)\}$ is given by the three-layered perceptron,

$$\varphi_i(w; x) = \rho(w_{i0} + \sum_{j=1}^{H} w_{ij} \, \rho(w_{j0} + \sum_{k=1}^{K} w_{jk} x_k)), \tag{7}$$

and $w = \{w_{i0}, w_{ij}\}$ is a set of biases and weights and $\rho(\cdot)$ is a sigmoidal function.

Let $M_{max}$ be the full-connected neural network model with $K$ input units, $H$ hidden units, and $L$ output units, and $\mathcal{M}$ be the family of all models made from $M_{max}$ by pruning weights or eliminating biases. When a set of training samples $\{(x_i, y_i)\}_{i=1}^{N}$ is given, we define an empirical loss and the prediction loss by

$$L_{emp}(w, \sigma) = -\frac{1}{N} \sum_{i=1}^{N} \log P(w, \sigma; y_i | x_i), \tag{8}$$

$$L(w, \sigma) = -\int \int Q(x, y) \log P(w, \sigma; y | x) dx dy. \tag{9}$$

Minimizing $L_{emp}(w, \sigma)$ is equivalent to minimizing $E_{emp}(w)$, and minimizing $L(w, \sigma)$ is equivalent to minimizing $E(w)$. We assume that there exists a parameter $(w_M^*, \sigma_M^*)$ which minimizes $L_{emp}(w, \sigma)$ in each model $M \in \mathcal{M}$. By the theory of AIC, we have the following formula,

$$< L(w_M^*, \sigma_M^*) > = < L_{emp}(w_M^*, \sigma_M^*) > + \frac{F(w_M^*) + 1}{N} + o(\frac{1}{N}). \tag{10}$$

Based on this property, let us define a formal information criterion $I(M)$ for a model $M$ by

$$I(M) = 2N L_{emp}(w_M^*, \sigma_M^*) + A(F_0(w_M^*) + 1) \tag{11}$$

where $A$ is a constant and $F_0(w)$ is the number of nonzero parameters in $w$,

$$F_0(w) = \sum_{i=1}^{L} \sum_{j=0}^{H} f_0(w_{ij}) + \sum_{j=1}^{H} \sum_{k=0}^{K} f_0(w_{jk}). \tag{12}$$

where $f_0(x)$ is 0 if $x = 0$, or 1 if otherwise. $I(M)$ is formally equal to AIC if $A = 2$, or MDL if $A = \log(N)$. Note that $F(w) \leq F_0(w)$ for arbitrary $w$ and that $F(w_M^*) = F_0(w_M^*)$ if and only if the Fisher information matrix of the model $M$ is positive definite.

## 2.2 A Modified Information Criterion

In order to find the optimal model and parameter simultaneously, we define a modified information criterion. For $\alpha > 0$,

$$I_\alpha(w, \sigma) = 2N L_{emp}(w, \sigma) + A(F_\alpha(w) + 1), \tag{13}$$

$$F_\alpha(w) = \sum_{i=1}^{L} \sum_{j=0}^{H} f_\alpha(w_{ij}) + \sum_{j=1}^{H} \sum_{k=0}^{K} f_\alpha(w_{jk}), \tag{14}$$

where $f_\alpha(x)$ satisfies the following two conditions.

(1) $f_\alpha(x) \to f_0(x)$ when $\alpha \to 0$.

(2) If $|x| \le |y|$ then $0 \le f_\alpha(x) \le f_\alpha(y) \le 1$.

For example, $1 - \exp(-x^2/\alpha^2)$ and $1 - 1/(1 + (x/\alpha)^2)$ satisfy this condition. Based on these definitions, we have the following theorem.

**Theorem** $$\min_{M \in \mathcal{M}} I(M) = \lim_{\alpha \to 0} \min_{w, \sigma} I_\alpha(w, \sigma).$$

This theorem shows that the optimal model and parameter can be found by minimizing $I_\alpha(w, \sigma)$ while controlling $\alpha$ as $\alpha \to 0$ (The parameter $\alpha$ plays the same role as the temperature in the simulated annealing). As $F_\alpha(x) \to F_0(x)$ is not uniform convergence, this theorem needs the second condition on $f_\alpha(x)$. (For proof of the theorem, see [5]).

If we choose a differentiable function for $f_\alpha(w)$, then its local minimum can be found by the steepest descent method,

$$\frac{dw}{dt} = -\frac{\partial}{\partial w} I_\alpha(w, \sigma), \qquad \frac{d\sigma}{dt} = -\frac{\partial}{\partial \sigma} I_\alpha(w, \sigma). \tag{15}$$

These equations result in a learning dynamics,

$$\Delta w = -\eta \sum_{i=1}^{N} \{\frac{\partial}{\partial w} \|y_i - \varphi(w; x_i)\|^2 + \frac{A\hat{\sigma}^2}{N} \frac{\partial F_\alpha}{\partial w}\}, \tag{16}$$

where $\hat{\sigma}^2 = (1/NL) \sum_{i=1}^{N} \|y_i - \varphi(w; x_i)\|^2$. and $\alpha$ is slowly controlled as $\alpha \to 0$. This dynamics can be understood as the error backpropagation with the added term.

## 3    Experimental Results

### 3.1    The true distribution is contained in the models

First, we consider a case when the true distribution is contained in the model family $\mathcal{M}$. Figure 1 (1) shows the true model from which the training samples were taken. One thousand input samples were taken from the uniform probability on $[-0.5, 0.5] \times [-0.5, 0.5] \times [-0.5, 0.5]$. The output samples were calculated by the network in Figure 1 (1), and noizes were added which were taken from a normal distribution with the expectation 0 and the variance $3.33 \times 10^{-3}$. Ten thousands testing samples were taken from the same distribution. We used $f_\alpha(w) = 1 - \exp(-w^2/2\alpha^2)$ as a softener function, and the "annealing schedule" of $\alpha$ was set as $\alpha(n) = \alpha_0(1 - n/n_{max}) + \epsilon$, where $n$ is the training cycle number, $\alpha_0 = 3.0$, $n_{max} = 25000$, and $\epsilon = 0.01$. Figure 1 (2) shows the full-connected model $M_{max}$ with 10 hidden units, which is the initial model. In the training, the learning speed $\eta$ was set as 0.1.

We compared the empirical errors and the prediction errors for several cases for $A$ (Figure 1 (5), (6)). If $A = 2$, the criterion is AIC, and if $A = \log(N) = 6.907$, it is BIC or MDL. Figure 1 (3) and (4) show the optimized models and parameters for the criteria with $A = 2$ and $A = 5$. When $A = 5$, the true model could be found.

## 3.2   The true distribution is not contained

Second, let us consider a case that the true distribution is *not* contained in the model family. For the training samples and the testing samples, we used the same probability density as the above case except that the function was

$$y = \frac{1}{4}\{\sin(\pi(x_1 + x_2)) + \tanh(x_3) + 2\}. \tag{17}$$

Figure 2 (1) and (2) show the training error and the prediction error, respectively. In this case, the best generalized model was found by AIC, shown in Figure 3. In the optimized network, $x_1$ and $x_2$ were almost separated from $x_3$, which means that the network could find the structure of the true model in eq.(17.)

The practical application to ultrasonic image reconstruction is shown in Figure 3.

# 4   Discussion

## 4.1   An information criterion and pruning weights

If $P(w, \sigma; y|x)$ sufficiently approximates $Q(y|x)$ and $N$ is sufficiently large, we have

$$L_{emp}(w_M^*, \sigma_M^*) = L(w_M^*, \sigma_M^*) + \frac{F(w_M^*) + 1}{N} + Z_N + o(\frac{1}{N}) \tag{18}$$

where $Z_N = L_{emp}(\hat{w}_M) - L(\hat{w}_M)$ and $\hat{w}_M$ is the parameter which minimizes $L(w, \sigma)$ in the model $M$. Although $< Z_N >= 0$ resulting in equation (10), its standard deviation has the same order as $(1/\sqrt{N})$. However, if $M_1 \subset M_2$ or $M_1 \supset M_2$, then $\hat{w}_{M1}$ and $\hat{w}_{M2}$ expected to be almost common, and it doesn't essentially affect the model selection problem [2].

The model family made by pruning weights or by eliminating biases is not a totally ordered set but a partially ordered set for the order "$\subset$". Therefore, if a model $M \in \mathcal{M}$ is selected, it is the optimal model in a local model family $\mathcal{M}' = \{M' \in \mathcal{M}; M' \subset M \text{ or } M' \supset M\}$, but it may not be the optimal model in the global family $\mathcal{M}$. Artificial neural networks have the local minimum problem not only in the parameter space but also in the model family.

## 4.2   The degenerate Fisher information matrix.

If the true probability is contained in the model and the number of hidden units is larger than necessary one, then the Fisher information matrix is degenerated, and consequently, the maximum likelihood estimator is not subject to the asymptotically normal distribution [6]. Therefore, the prediction error is not given by eq.(3), or AIC cannot be derived. However, by the proposed method, the selected model has the non-degenerated Fiher information matrix, because if it is degenerate then the modified information criterion is not minimized.

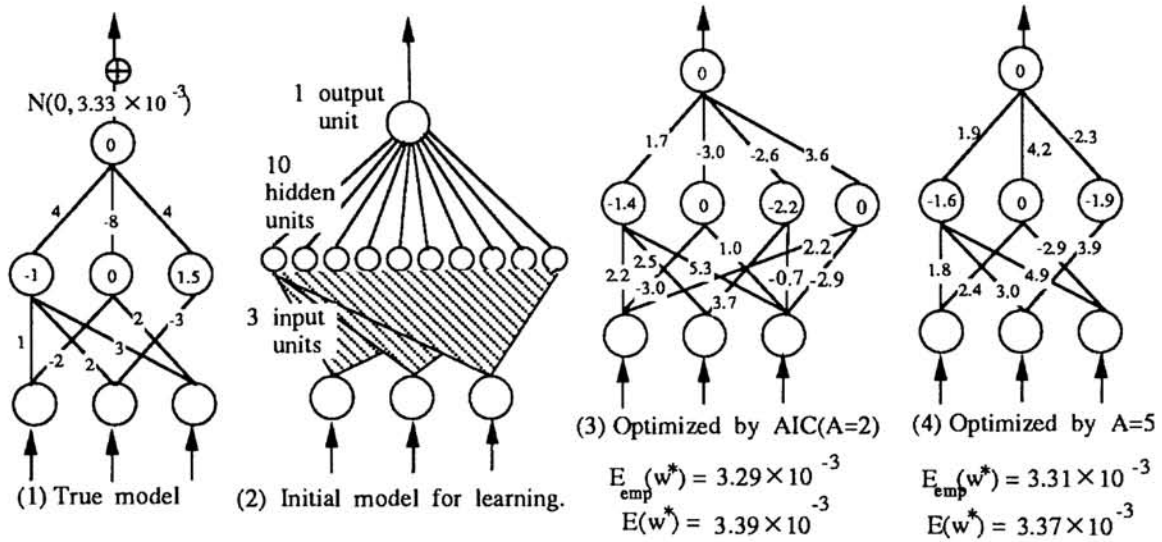

(1) True model

(2) Initial model for learning.

(3) Optimized by AIC(A=2)

$E_{emp}(w^*) = 3.29 \times 10^{-3}$

$E(w^*) = 3.39 \times 10^{-3}$

(4) Optimized by A=5

$E_{emp}(w^*) = 3.31 \times 10^{-3}$

$E(w^*) = 3.37 \times 10^{-3}$

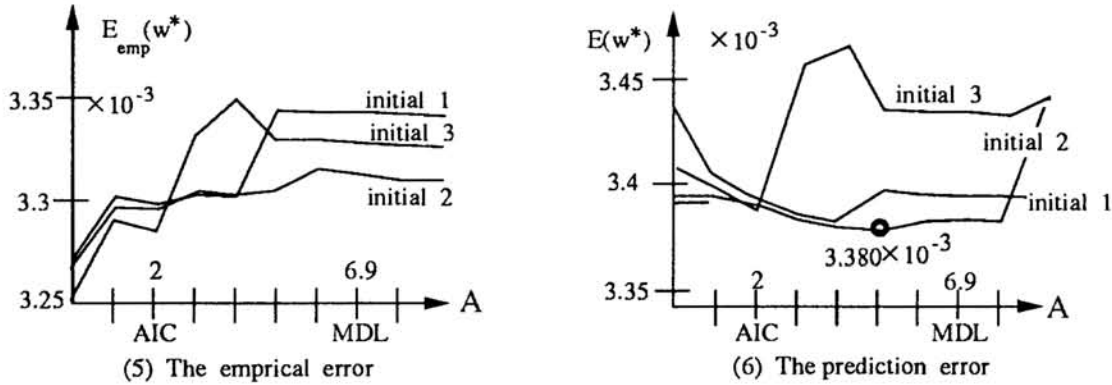

(5) The emprical error

(6) The prediction error

Figure 1:  True distribution is contained in the models.

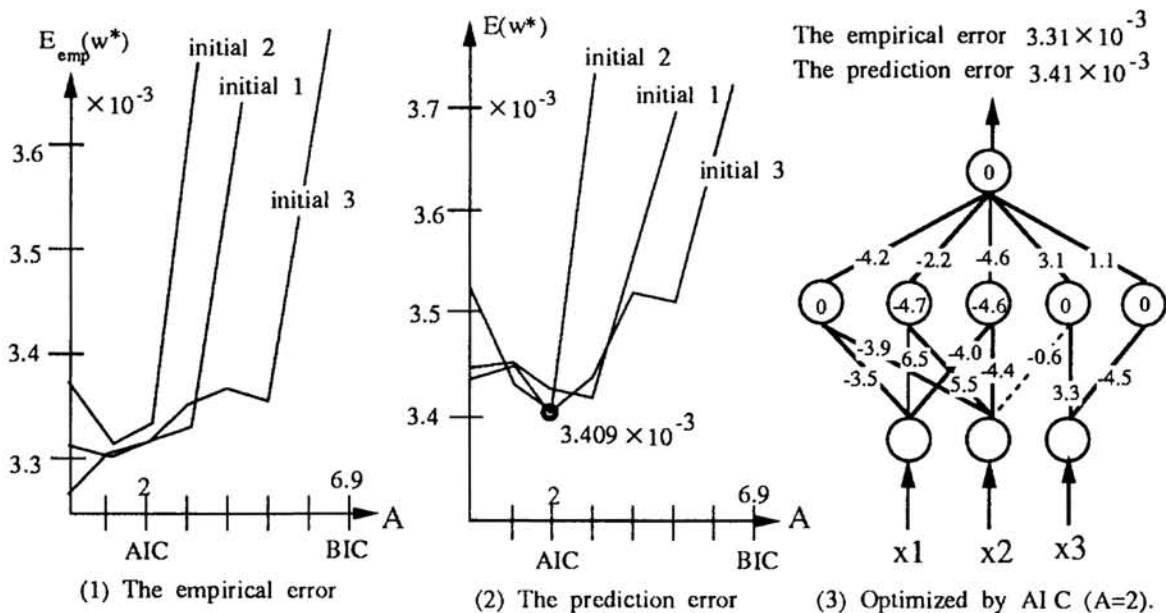

(1) The empirical error

(2) The prediction error

The empirical error  $3.31 \times 10^{-3}$

The prediction error  $3.41 \times 10^{-3}$

(3) Optimized by AI C (A=2).

Figure 2:  True distribution is *not* contained in the models.

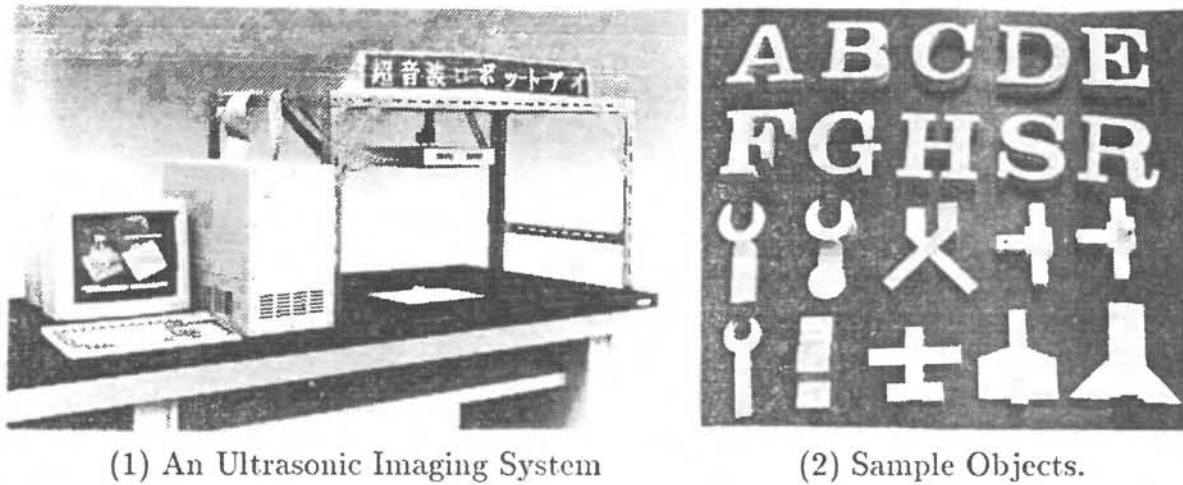

(1) An Ultrasonic Imaging System          (2) Sample Objects.

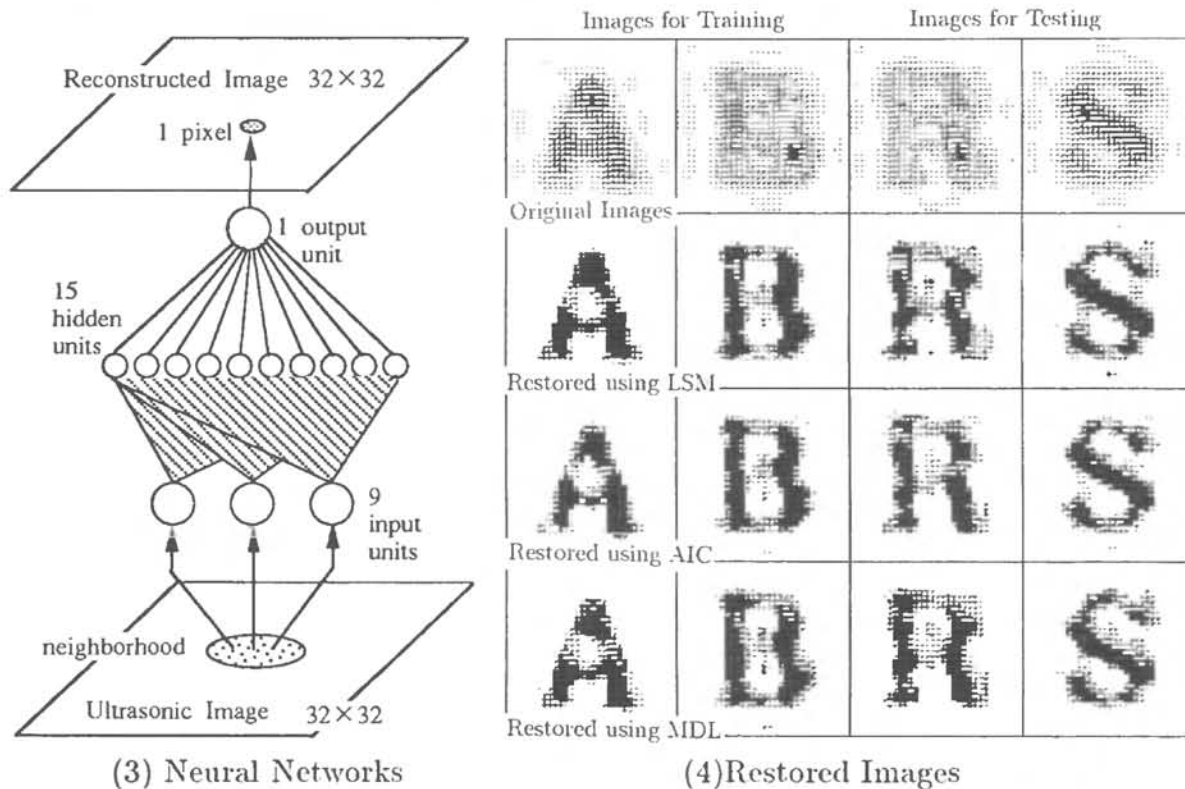

(3) Neural Networks          (4) Restored Images

Figure 3: Practical Application to Image Restoration

The proposed method was applied to ultrasonic image restoration. Figure 3 (1), (2), (3), (4) respectively show an ultrasonic imaging system, the sample objects, and a neural network for image restoration, and the original restored images. The number of parameters optimized by LSM, AIC, and MDL were respectively 166, 138, and 57. Rather noizeless images were obtained using the modified AIC or MDL. For example, the "Tail of R" was clearly restored using AIC.

### 4.3    Relation to another generalization methods

In the neural information processing field, many methods have been proposed for preventing the over-fitting problem. One of the most famous methods is the weight decay method, in which we assume a priori probability distribution on the parameter space and minimize

$$E_1(w) = E_{emp}(w) + \lambda C(w), \tag{19}$$

where $\lambda$ and $C(w)$ are chosen by several heuristic methods [7]. The BIC is the information criterion for such a method [3], and the proposed method may be understood as a method how to control $\lambda$ and $C(w)$.

## 5    Conclusion

An optimization method for layered neural networks was proposed based on the modified information criterion, and its effectiveness was discussed theoretically and experimentally.

### Acknowledgements

The author would like to thank Prof. S. Amari, Prof. S. Yoshizawa, Prof. K. Aihara in University of Tokyo, and all members of the Amari seminar for their active discussions about statistical methods in neural networks.

### References

[1] H.Akaike. (1974) A New Look at the Statistical Model Identification. it IEEE Trans. on Automatic Control, Vol.AC-19, No.6, pp.716-723.

[2] N.Murata, S.Yoshizawa, and S.Amari.(1992) Learning Curves, Model Selection and Complexity of Neural Networks. *Advances in Neural Information Processing Systems 5*, San Mateo, Morgan Kaufman, pp.607-614.

[3] C.Schwarz (1978) Estimating the dimension of a model. *Annals of Statistics* Vol.6, pp.461-464.

[4] J.Rissanen. (1984) Universal Coding, Information, Prediction, and Estimation. *IEEE Trans. on Information Theory*, Vol.30, pp.629-636.

[5] S.Watanabe. (1993) An Optimization Method of Artificial Neural Networks based on a Modified Information Criterion. *IEICE technical Report* Vol.NC93-52, pp.71-78.

[6] H.White. (1989) Learning in Artificial Neural Networks : A Statistical Perspective. *Neural Computation*, Vol.1, pp.425-464.

[7] A.S.Weigend, D.E.Rumelhart, and B.A.Huberman. (1991) Generalization of weight-elimination with application to forecasting. *Advances in Neural Information Processing Systems*, Vol.3, pp.875-882.
